# Convergence of Stochastic Iterative Dynamic Programming Algorithms

**Tommi Jaakkola**[*]        **Michael I. Jordan**
**Satinder P. Singh**
Department of Brain and Cognitive Sciences
Massachusetts Institute of Technology
Cambridge, MA 02139

## Abstract

Increasing attention has recently been paid to algorithms based on dynamic programming (DP) due to the suitability of DP for learning problems involving control. In stochastic environments where the system being controlled is only incompletely known, however, a unifying theoretical account of these methods has been missing. In this paper we relate DP-based learning algorithms to the powerful techniques of stochastic approximation via a new convergence theorem, enabling us to establish a class of convergent algorithms to which both TD($\lambda$) and Q-learning belong.

## 1   INTRODUCTION

Learning to predict the future and to find an optimal way of controlling it are the basic goals of learning systems that interact with their environment. A variety of algorithms are currently being studied for the purposes of prediction and control in incompletely specified, stochastic environments. Here we consider learning algorithms defined in Markov environments. There are actions or controls ($u$) available for the learner that affect both the state transition probabilities, and the probability distribution for the immediate, state dependent costs ($c_i(u)$) incurred by the learner. Let $p_{ij}(u)$ denote the probability of a transition to state $j$ when control $u$ is executed in state $i$. The learning problem is to predict the expected cost of a

---

[*]E-mail: tommi@psyche.mit.edu

fixed policy $\mu$ (a function from states to actions), or to obtain the optimal policy ($\mu^*$) that minimizes the expected cost of interacting with the environment.

If the learner were allowed to know the transition probabilities as well as the immediate costs the control problem could be solved directly by Dynamic Programming (see e.g., Bertsekas, 1987). However, when the underlying system is only incompletely known, algorithms such as Q-learning (Watkins, 1989) for prediction and control, and TD($\lambda$) (Sutton, 1988) for prediction, are needed.

One of the central problems in developing a theoretical understanding of these algorithms is to characterize their convergence; that is, to establish under what conditions they are ultimately able to obtain correct predictions or optimal control policies. The stochastic nature of these algorithms immediately suggests the use of stochastic approximation theory to obtain the convergence results. However, there exists no directly available stochastic approximation techniques for problems involving the maximum norm that plays a crucial role in learning algorithms based on DP.

In this paper, we extend Dvoretzky's (1956) formulation of the classical Robbins-Munro (1951) stochastic approximation theory to obtain a class of converging processes involving the maximum norm. In addition, we show that Q-learning and both the on-line and batch versions of TD($\lambda$) are realizations of this new class. This approach keeps the convergence proofs simple and does not rely on constructions specific to particular algorithms. Several other authors have recently presented results that are similar to those presented here: Dayan and Sejnowski (1993) for TD($\lambda$), Peng and Williams (1993) for TD($\lambda$), and Tsitsiklis (1993) for Q-learning. Our results appear to be closest to those of Tsitsiklis (1993).

## 2   Q-LEARNING

The Q-learning algorithm produces values—"Q-values"—by which an optimal action can be determined at any state. The algorithm is based on DP by rewriting Bellman's equation such that there is a value assigned to every state-action pair instead of only to a state. Thus the Q-values satisfy

$$Q(s, u) = \bar{c}_s(u) + \gamma \sum_{s'} p_{ss'}(u) \max_{u'} Q(s', u') \tag{1}$$

where $\bar{c}$ denotes the mean of $c$. The solution to this equation can be obtained by updating the Q-values iteratively; an approach known as the *value iteration* method. In the learning problem the values for the mean of $c$ and for the transition probabilities are unknown. However, the observable quantity

$$c_{s_t}(u_t) + \gamma \max_u Q(s_{t+1}, u) \tag{2}$$

where $s_t$ and $u_t$ are the state of the system and the action taken at time $t$, respectively, is an unbiased estimate of the update used in value iteration. The Q-learning algorithm is a relaxation method that uses this estimate iteratively to update the current Q-values (see below).

The Q-learning algorithm converges mainly due to the contraction property of the value iteration operator.

## 2.1   CONVERGENCE OF Q-LEARNING

Our proof is based on the observation that the Q-learning algorithm can be viewed as a stochastic process to which techniques of stochastic approximation are generally applicable. Due to the lack of a formulation of stochastic approximation for the maximum norm, however, we need to slightly extend the standard results. This is accomplished by the following theorem the proof of which can be found in Jaakkola *et al.* (1993).

**Theorem 1** *A random iterative process $\Delta_{n+1}(x) = (1-\alpha_n(x))\Delta_n(x)+\beta_n(x)F_n(x)$ converges to zero w.p.1 under the following assumptions:*

  *1) The state space is finite.*

  *2) $\sum_n \alpha_n(x) = \infty$, $\sum_n \alpha_n^2(x) < \infty$, $\sum_n \beta_n(x) = \infty$, $\sum_n \beta_n^2(x) < \infty$, and $\mathrm{E}\{\beta_n(x)|P_n\} \leq \mathrm{E}\{\alpha_n(x)|P_n\}$ uniformly w.p.1.*

  *3) $\| \mathrm{E}\{F_n(x)|P_n\} \|_W \leq \gamma \| \Delta_n \|_W$, where $\gamma \in (0,1)$.*

  *4) $\mathrm{Var}\{F_n(x)|P_n\} \leq C(1+ \| \Delta_n \|_W)^2$, where $C$ is some constant.*

*Here $P_n = \{\Delta_n, \Delta_{n-1}, \ldots, F_{n-1}, \ldots, \alpha_{n-1}, \ldots, \beta_{n-1}, \ldots\}$ stands for the past at step n. $F_n(x)$, $\alpha_n(x)$ and $\beta_n(x)$ are allowed to depend on the past insofar as the above conditions remain valid. The notation $\| \cdot \|_W$ refers to some weighted maximum norm.*

In applying the theorem, the $\Delta_n$ process will generally represent the difference between a stochastic process of interest and some optimal value (e.g., the optimal value function). The formulation of the theorem therefore requires knowledge to be available about the optimal solution to the learning problem before it can be applied to any algorithm whose convergence is to be verified. In the case of Q-learning the required knowledge is available through the theory of DP and Bellman's equation in particular.

The convergence of the Q-learning algorithm now follows easily by relating the algorithm to the converging stochastic process defined by Theorem 1.[1]

**Theorem 2** *The Q-learning algorithm given by*

$$Q_{t+1}(s_t, u_t) = (1 - \alpha_t(s_t, u_t))Q_t(s_t, u_t) + \alpha_t(s_t, u_t)[c_{s_t}(u_t) + \gamma V_t(s_{t+1})]$$

*converges to the optimal $Q^*(s, u)$ values if*

  *1) The state and action spaces are finite.*

  *2) $\sum_t \alpha_t(s, u) = \infty$ and $\sum_t \alpha_t^2(s, u) < \infty$ uniformly w.p.1.*

  *3) $\mathrm{Var}\{c_s(u)\}$ is bounded.*

*3) If $\gamma = 1$, all policies lead to a cost free terminal state w.p.1.*

**Proof.** By subtracting $Q^*(s, u)$ from both sides of the learning rule and by defining $\Delta_t(s, u) = Q_t(s, u) - Q^*(s, u)$ together with

$$F_t(s, u) = c_s(u) + \gamma V_t(s_{next}) - Q^*(s, u) \tag{3}$$

the Q-learning algorithm can be seen to have the form of the process in Theorem 1 with $\beta_t(s, u) = \alpha_t(s, u)$.

To verify that $F_t(s, u)$ has the required properties we begin by showing that it is a contraction mapping with respect to some maximum norm. This is done by relating $F_t$ to the DP value iteration operator for the same Markov chain. More specifically,

$$
\begin{aligned}
\max_u |E\{F_t(i, u)\}| &= \gamma \max_u \left| \sum_j p_{ij}(u)[V_t(j) - V^*(j)] \right| \\
&\leq \gamma \max_u \sum_j p_{ij}(u) \max_v |Q_t(j, v) - Q^*(j, v)| \\
&= \gamma \max_u \sum_j p_{ij}(u) V^\Delta(j) = T(V^\Delta)(i)
\end{aligned}
$$

where we have used the notation $V^\Delta(j) = \max_v |Q_t(j, v) - Q^*(j, v)|$ and $T$ is the DP value iteration operator for the case where the costs associated with each state are zero. If $\gamma < 1$ the contraction property of $E\{F_t(i, u)\}$ can be obtained by bounding $\sum_j p_{ij}(u) V^\Delta(j)$ by $\max_j V^\Delta(j)$ and then including the $\gamma$ factor. When the future costs are not discounted ($\gamma = 1$) but the chain is absorbing and all policies lead to the terminal state w.p.1 there still exists a weighted maximum norm with respect to which $T$ is a contraction mapping (see e.g. Bertsekas & Tsitsiklis, 1989) thereby forcing the contraction of $E\{F_t(i, u)\}$. The variance of $F_t(s, u)$ given the past is within the bounds of Theorem 1 as it depends on $Q_t(s, u)$ at most linearly and the variance of $c_s(u)$ is bounded.

Note that the proof covers both the on-line and batch versions.  □

## 3   THE TD($\lambda$) ALGORITHM

The TD($\lambda$) (Sutton, 1988) is also a DP-based learning algorithm that is naturally defined in a Markov environment. Unlike Q-learning, however, TD does not involve decision-making tasks but rather predictions about the future costs of an evolving system. TD($\lambda$) converges to the same predictions as a version of Q-learning in which there is only one action available at each state, but the algorithms are derived from slightly different grounds and their behavioral differences are not well understood.

The algorithm is based on the estimates

$$V_t^\lambda(i) = (1 - \lambda) \sum_{n=1}^\infty \lambda^{n-1} V_t^{(n)}(i) \tag{4}$$

where $V_t^{(n)}(i)$ are $n$ step look-ahead predictions. The expected values of the $V_t^\lambda(i)$ are strictly better estimates of the correct predictions than the $V_t(i)$s are (see

Jaakkola et al., 1993) and the update equation of the algorithm

$$V_{t+1}(i_t) = V_t(i_t) + \alpha_t[V_t^\lambda(i_t) - V_t(i_t)] \tag{5}$$

can be written in a practical recursive form as is seen below. The convergence of the algorithm is mainly due to the statistical properties of the $V_t^\lambda(i)$ estimates.

## 3.1   CONVERGENCE OF TD($\lambda$)

As we are interested in strong forms of convergence we need to impose some new constraints, but due to the generality of the approach we can dispense with some others. Specifically, the learning rate parameters $\alpha_n$ are replaced by $\alpha_n(i)$ which satisfy $\sum_n \alpha_n(i) = \infty$ and $\sum_n \alpha_n^2(i) < \infty$ uniformly w.p.1. These parameters allow asynchronous updating and they can, in general, be random variables. The convergence of the algorithm is guaranteed by the following theorem which is an application of Theorem 1.

**Theorem 3** *For any finite absorbing Markov chain, for any distribution of starting states with no inaccessible states, and for any distributions of the costs with finite variances the TD($\lambda$) algorithm given by*

*1)*

$$V_{n+1}(i) = V_n(i) + \alpha_n(i) \sum_{t=1}^{m} [c_{i_t} + \gamma V_n(i_{t+1}) - V_n(i_t)] \sum_{k=1}^{t} (\gamma\lambda)^{t-k} \chi_i(k)$$

$\sum_n \alpha_n(i) = \infty$ *and* $\sum_n \alpha_n^2(i) < \infty$ *uniformly w.p.1.*

*2)*

$$V_{t+1}(i) = V_t(i) + \alpha_t(i)[c_{i_t} + \gamma V_t(i_{t+1}) - V_t(i_t)] \sum_{k=1}^{t} (\gamma\lambda)^{t-k} \chi_i(k)$$

$\sum_t \alpha_t(i) = \infty$ *and* $\sum_n \alpha_t^2(i) < \infty$ *uniformly w.p.1 and within sequences* $\alpha_t(i)/max_{t \in S}\alpha_t(i) \to 1$ *uniformly w.p.1.*

*converges to the optimal predictions w.p.1 provided* $\gamma, \lambda \in [0,1]$ *with* $\gamma\lambda < 1$.

**Proof for (1):** We use here a slightly different form for the learning rule (cf. the previous section).

$$V_{n+1}(i) = V_n(i) + \alpha_n(i)[G_n(i) - \frac{m(i)}{E\{m(i)\}} V_n(i)]$$

$$G_n(i) = \frac{1}{E\{m(i)\}} \sum_{k=1}^{m(i)} V_n^\lambda(i;k)$$

where $V_n^\lambda(i;k)$ is an estimate calculated at the $k^{th}$ occurrence of state $i$ in a sequence and for mathematical convenience we have made the transformation $\alpha_n(i) \to E\{m(i)\}\alpha_n(i)$, where $m(i)$ is the number of times state $i$ was visited during the sequence.

To apply Theorem 1 we subtract $V^*(i)$, the optimal predictions, from both sides of the learning equation. By identifying $\alpha_n(i) := \alpha_n(i)m(i)/\mathrm{E}\{m(i)\}$, $\beta_n(i) := \alpha_n(i)$, and $F_n(i) := G_n(i) - V^*(i)m(i)/\mathrm{E}\{m(i)\}$ we need to show that these satisfy the conditions of Theorem 1. For $\alpha_n(i)$ and $\beta_n(i)$ this is obvious. We begin here by showing that $F_n(i)$ indeed is a contraction mapping. To this end,

$$\max_i |\mathrm{E}\{F_n(i) \mid V_n\}| =$$

$$\max_i |\frac{1}{\mathrm{E}\{m(i)\}}\mathrm{E}\{(V_n^\lambda(i;1) - V^*(i)) + (V_n^\lambda(i;2) - V^*(i)) + \ldots \mid V_n\}|$$

which can be bounded above by using the relation

$$|\mathrm{E}\{V_n^\lambda(i;k) - V^*(i) \mid V_n\}|$$
$$\leq \quad \mathrm{E}\left\{ |\mathrm{E}\{V_n^\lambda(i;k) - V^*(i) \mid m(i) \geq k, V_n\}|\theta(m(i) - k) \mid V_n\right\}$$
$$\leq \quad P\{m(i) \geq k\}|\mathrm{E}\{V_n^\lambda(i) - V^*(i) \mid V_n\}|$$
$$\leq \quad \gamma P\{m(i) \geq k\} \max_i |V_n(i) - V^*(i)|$$

where $\theta(x) = 0$ if $x < 0$ and 1 otherwise. Here we have also used the fact that $V_n^\lambda(i)$ is a contraction mapping independent of possible discounting. As $\sum_k P\{m(i) \geq k\} = \mathrm{E}\{m(i)\}$ we finally get

$$\max_i |\mathrm{E}\{F_n(i) \mid V_n\}| \leq \gamma \max_i |V_n(i) - V^*(i)|$$

The variance of $F_n(i)$ can be seen to be bounded by

$$\mathrm{E}\{m^4\} \max_i |V_n(i)|^2$$

For any absorbing Markov chain the convergence to the terminal state is geometric and thus for every finite $k$, $\mathrm{E}\{m^k\} \leq C(k)$, implying that the variance of $F_n(i)$ is within the bounds of Theorem 1. As Theorem 1 is now applicable we can conclude that the batch version of TD($\lambda$) converges to the optimal predictions w.p.1.   $\square$

**Proof for (2)** The proof for the on-line version is achieved by showing that the effect of the on-line updating vanishes in the limit thereby forcing the two versions to be equal asymptotically. We view the on-line version as a batch algorithm in which the updates are made after each complete sequence but are made in such a manner so as to be equal to those made on-line.

Define $G_n'(i) = G_n(i) + G_n^\Delta(i)$ to be a new batch estimate taking into account the on-line updating within sequences. Here $G_n(i)$ is the batch estimate with the desired properties (see the proof for (1)) and $G_n^\Delta(i)$ is the difference between the two. We take the new batch learning parameters to be the maxima over a sequence, that is $a_n(i) = \max_{t \in S} \alpha_t(i)$. As all the $\alpha_t(i)$ satisfy the required conditions uniformly w.p.1 these new learning parameters satisfy them as well.

To analyze the new batch algorithm we divide it into three parallel processes: the batch TD($\lambda$) with $a_n(i)$ as learning rate parameters, the difference between this and the new batch estimate, and the change in the value function due to the updates made on-line. Under the conditions of the TD($\lambda$) convergence theorem rigorous

upper bounds can be derived for the latter two processes (see Jaakkola, et al., 1993). These results enable us to write

$$
\begin{aligned}
\| \, \mathrm{E}\{G_n' - V^*\} \, \| \; &\leq \; \| \, \mathrm{E}\{G_n - V^*\} \, \| + \| \, G_n^\Delta \, \| \\
&\leq \; (\gamma' + C_n^1) \, \| \, V_n - V^* \, \| + C_n^2
\end{aligned}
$$

where $C_n^1$ and $C_n^2$ go to zero with w.p.1. This implies that for any $\epsilon > 0$ and $\| \, V_n - V^* \, \| \gg \epsilon$ there exists $\gamma < 1$ such that

$$
\| \, \mathrm{E}\{G_n' - V^*\} \, \| \leq \gamma \, \| \, V_n - V^* \, \|
$$

for $n$ large enough. This is the required contraction property of Theorem 1. In addition, it can readily be checked that the variance of the new estimate falls under the conditions of Theorem 1.

Theorem 1 now guarantees that for any $\epsilon$ the value function in the on-line algorithm converges w.p.1 into some $\epsilon$-bounded region of $V^*$ and therefore the algorithm itself converges to $V^*$ w.p.1. □

## 4   CONCLUSIONS

In this paper we have extended results from stochastic approximation theory to cover asynchronous relaxation processes which have a contraction property with respect to some maximum norm (Theorem 1). This new class of converging iterative processes is shown to include both the Q-learning and TD($\lambda$) algorithms in either their on-line or batch versions. We note that the convergence of the on-line version of TD($\lambda$) has not been shown previously. We also wish to emphasize the simplicity of our results. The convergence proofs for Q-learning and TD($\lambda$) utilize only high-level statistical properties of the estimates used in these algorithms and do not rely on constructions specific to the algorithms. Our approach also sheds additional light on the similarities between Q-learning and TD($\lambda$).

Although Theorem 1 is readily applicable to DP-based learning schemes, the theory of Dynamic Programming is important only for its characterization of the optimal solution and for a contraction property needed in applying the theorem. The theorem can be applied to iterative algorithms of different types as well.

Finally we note that Theorem 1 can be extended to cover processes that do not show the usual contraction property thereby increasing its applicability to algorithms of possibly more practical importance.

## Footnotes

[1]We note that the theorem is more powerful than is needed to prove the convergence of Q-learning. Its generality, however, allows it to be applied to other algorithms as well (see the following section on TD($\lambda$)).

## References

Bertsekas, D. P. (1987). *Dynamic Programming: Deterministic and Stochastic Models*. Englewood Cliffs, NJ: Prentice-Hall.

Bertsekas, D. P., & Tsitsiklis, J. N. (1989). *Parallel and Distributed Computation: Numerical Methods*. Englewood Cliffs, NJ: Prentice-Hall.

Dayan, P. (1992). The convergence of TD($\lambda$) for general $\lambda$. *Machine Learning, 8*, 341-362.

Dayan, P., & Sejnowski, T. J. (1993). *TD(λ) converges with probability 1.* CNL, The Salk Institute, San Diego, CA.

Dvoretzky, A. (1956). On stochastic approximation. *Proceedings of the Third Berkeley Symposium on Mathematical Statistics and Probability.* University of California Press.

Jaakkola, T., Jordan, M. I., & Singh, S. P. (1993). On the convergence of stochastic iterative dynamic programming algorithms. Submitted to *Neural Computation.*

Peng J., & Williams R. J. (1993). TD(λ) converges with probability 1. Department of Computer Science preprint, Northeastern University.

Robbins, H., & Monro, S. (1951). A stochastic approximation model. *Annals of Mathematical Statistics, 22*, 400-407.

Sutton, R. S. (1988). Learning to predict by the methods of temporal differences. *Machine Learning, 3*, 9-44.

Tsitsiklis J. N. (1993). Asynchronous stochastic approximation and Q-learning. Submitted to: *Machine Learning.*

Watkins, C.J.C.H. (1989). *Learning from delayed rewards.* PhD Thesis, University of Cambridge, England.

Watkins, C.J.C.H, & Dayan, P. (1992). Q-learning. *Machine Learning, 8*, 279-292.
